# Local Bandit Approximation
# for Optimal Learning Problems

Michael O. Duff      Andrew G. Barto
Department of Computer Science
University of Massachusetts
Amherst, MA 01003
{duff,barto}@cs.umass.edu

## Abstract

In general, procedures for determining Bayes-optimal adaptive controls for Markov decision processes (MDP's) require a prohibitive amount of computation—the optimal learning problem is intractable. This paper proposes an approximate approach in which bandit processes are used to model, in a certain "local" sense, a given MDP. Bandit processes constitute an important subclass of MDP's, and have optimal learning strategies (defined in terms of Gittins indices) that can be computed relatively efficiently. Thus, one scheme for achieving approximately-optimal learning for general MDP's proceeds by taking actions suggested by strategies that are optimal with respect to local bandit models.

## 1 INTRODUCTION

Watkins [1989] has defined *optimal learning* as: "... the process of collecting and using information during learning in an optimal manner, so that the learner makes the best possible decisions at all stages of learning: learning itself is regarded as a multistage decision process, and learning is optimal if the learner adopts a strategy that will yield the highest possible return from actions over the whole course of learning."

For example, suppose a decision-maker is presented with two biased coins (the decision-maker does not know precisely how the coins are biased) and asked to allocate twenty flips between them so as to maximize the number of observed heads. Although the decision-maker is certainly interested in determining which coin has a higher probability of heads, his principle concern is with optimizing performance *en route* to this determination. An optimal learning strategy typically intersperses "exploitation" steps, in which the coin currently thought to have the highest proba-

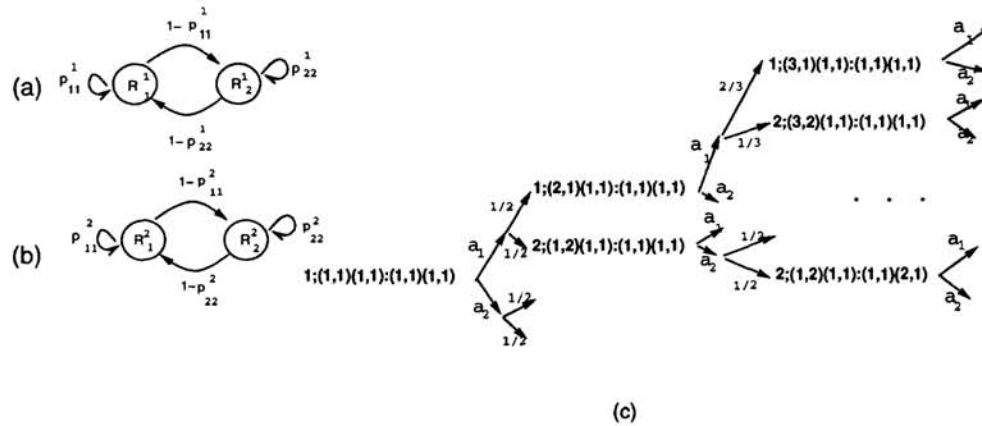

Figure 1: A simple example: dynamics/rewards under (a) action 1 and (b) action 2. (c) The decision problem in hyperstate space.

bility of heads is flipped, with "exploration" steps in which, on the basis of observed flips, a coin that would be deemed inferior is flipped anyway to further resolve its true potential for turning up heads. The coin-flip problem is a simple example of a (two-armed) *bandit problem*. A key feature of these problems, and of adaptive control processes in general, is the so-called "exploration-versus-exploitation trade-off" (or problem of "dual control" [Fel'dbaum, 1965]).

As an another example, consider the MDP depicted in Figures 1(a) and (b). This is a 2-state/2-action process; transition probabilities label arcs, and quantities within circles denote expected rewards for taking particular actions in particular states. The goal is to assign actions to states so as to maximize, say, the expected infinite horizon discounted sum of rewards (the value function) over all states. For the case considered in this paper, the transition probabilites are not known. Given that the process is in some state, one action may be optimal with respect to currently-perceived point-estimates of unknown parameters, while another action may result in greater information gain. Optimal learning is concerned with striking a balance between these two criteria.

While reinforcement learning approaches *have* recognized the dual-effects of control, at least in the sense that one must occasionally deviate from a greedy policy to ensure a search of sufficient breadth, many exploration procedures appear not to be motivated by real notions of optimal learning; rather, they aspire to be practical schemes for avoiding unrealistic levels of sampling and search that would be required if one were to strictly adhere to the theoretical sufficient conditions for convergence—that all state-action pairs must be considered infinitely many times.

If one is willing to adopt a Bayesian perspective, then the exploration-versus-exploitation issue has already been resolved, in principle. A solution was recognized by Bellman and Kalaba nearly fo rty years ago [Bellman & Kalaba, 1959]; their dynamic programming algorithm for computing Bayes-optimal policies begins by regarding "state" as an ordered pair, or "hyperstate," $(x, \mathcal{I})$, where $x$ is a point in phase-space (Markov-chain state) and $\mathcal{I}$ is the "information pattern," which summarizes past history as it relates to modeling the transitional dynamics of $x$. Computation grows increasingly burdensome with problem size, however, so one is compelled to seek approximate solutions, some of which ignore the effects of information gain entirely. In contrast, the approach suggested in this paper explicitly acknowledges that there is an information-gain component to the optimal learn-

ing problem; if certain salient aspects of the value of information can be captured, even approximately, then one may be led to a reasonable method for approximating optimal learning policies.

Here is the basic idea behind the approach suggested in this paper: First note that there exists a special class of problems, namely multi-armed bandit problems, in which the information pattern is the sole component of the hyperstate. These special problems have the important feature that their optimal policies can be defined concisely in terms of "Gittins indices," and these indices can be computed in a relatively efficient way. This paper is an attempt to make use of the fact that this special subclass of MDP's has tractably-computable optimal learning strategies. Actions for general MDP's are derived by, first, attaching to a given general MDP in a given state a "local" n-armed bandit process that captures some aspect of the value of information gain as well as explicit reward. Indices for the local bandit model can be computed relatively efficiently; the largest index suggests the best action in an optimal-learning sense. The resulting algorithm has a receding-horizon flavor in that a new local-bandit process is constructed after each transition; it makes use of a mean-process model as in some previously-suggested approximation schemes, but here the value of information gain is explicitly taken into account, in part, through index calculations.

## 2 THE BAYES-BELLMAN APPROACH FOR ADAPTIVE MDP'S

Consider the two-state, two-action process shown in Figure 1, and suppose that one is uncertain about the transition probabilities. If the process is in a given state and an action is taken, then the result is that the process either stays in the state it is in or jumps to the other state—one observes a Bernoulli process with unknown parameter—just as in the coin-flip example. But in this case one observes *four* Bernoulli processes: the result of taking action 1 in state 1, action 1 in state 2, action 2 in state 1, action 2 in state 2. So if the prior probability for staying in the current state, for each of these state-action pairs, is represented by a beta distribution (the appropriate conjugate family of distributions with regard to Bernoulli sampling; i.e., a Bayesian update of a beta prior remains beta), then one may perform dynamic programming in a space of "hyperstates," in which the components are four pairs of parameters specifying the beta distributions describing the uncertainty in the transition probabilities, along with the Markov chain state: $\langle x, (\alpha_1^1, \beta_1^1), (\alpha_2^1, \beta_2^1)(\alpha_1^2, \beta_1^2), (\alpha_2^2, \beta_2^2)\rangle$, where for example $(\alpha_1^1, \beta_1^1)$ denotes the parmeters specifying the beta distribution that represents uncertainty in the transition probability $p_{11}^1$. Figure 1(c) shows part of the associated decision tree; an optimality equation may be written in terms of the hyperstates. MDP's with more than two states pose no special problem (there exists an appropriate generalization of the beta distribution). What *is* a problem is what Bellman calls the "problem of the expanding grid:" the number of hyperstates that must be examined grows exponentially with the horizon.

How does one proceed if one is constrained to practical amounts of computation and is willing to settle for an approximate solution? One could truncate the decision tree at some shorter and more manageable horizon, compute approximate terminal values by replacing the distributions with their means, and proceed with a receding-horizon approach: Starting from the approximate terminal values at the horizon, perform a backward sweep of dynamic programming, computing an optimal policy. Take the initial action of the policy, then shift the entire computational window forward one level and repeat. One can imagine a sort of limiting, degenerate version

of this receding horizon approach in which the horizon is zero; that is, use the means of the current distributions to calculate an optimal policy, take an "optimal" action, observe a transition, perform a Bayesian modification of the prior, and repeat. This (certainty-equivalence) heuristic was suggested by [Cozzolino et al., 1965], and has recently reappeared in [Dayan & Sejnowski, 1996]. However, as was noted in [Cozzolino et al., 1965] "...the trade-off between immmediate gain and information does not exist in this heuristic. There is no mechanism which explicitly forces unexplored policies to be observed in early stages. Therefore, if it should happen that there is some very good policy which *a priori* seemed quite bad, it is entirely possible that this heuristic will never provide the information needed to recognize the policy as being better than originally thought..." This comment and others seem to refer to what is now regarded as a problem of "identifiability" associated with certainty-equivalence controllers in which a closed-loop system evolves identically for both true and false values of the unknown parameters; that is, certainty-equivalence control may make some of the unknown parameters invisible to the identification process and lead one to repeatedly choose the wrong action (see [Borkar & Varaiya, 1979], and also Watkins' discussion of "metastable policies" in [Watkins, 1989]).

## 3   BANDIT PROBLEMS AND INDEX COMPUTATION

One basic version of the bandit problem may be described as follows: There are some number of statistically independent reward processes—Markov chains with an imposed reward structure associated with the chain's arcs. At each discrete time-step, a decision-maker selects one of these processes to activate. The activated process yields an immediate reward and then changes state. The other processes remain frozen and yield no reward. The goal is to splice together the individual reward streams into one sequence having maximal expected discounted value.

The special Cartesian structure of the bandit problem turns out to imply that there are functions that map process-states to scalars (or "indices"), such that optimal policies consist simply of activating the task with the largest index. Consider one of the reward processes, let $S$ be its state space, and let $B$ be the set of all subsets of $S$. Suppose that $x(k)$ is the state of the process at time $k$ and, for $B \in B$, let $\tau(B)$ be the number of transitions until the process first enters the set $B$. Let $\nu(i; B)$ be the expected discounted reward per unit of discounted time starting from state $i$ until the stopping time $\tau(B)$:

$$\nu(i; B) = \frac{E\left\{\sum_{k=0}^{\tau(B)-1} \gamma^k R(x(k)) | x(0) = i\right\}}{E\left\{\sum_{k=0}^{\tau(B)-1} \gamma^k | x(0) = i\right\}}.$$

Then the Gittins index of state $i$ for the process under consideration is

$$\nu(i) = \max_{B \in B} \nu(i; B). \tag{1}$$

[Gittins & Jones, 1979] shows that the indices may be obtained by solving a set of functional equations. Other algorithms that have been suggested include those by Beale (see the discussion section following [Gittins & Jones, 1979]), [Robinsion, 1981], [Varaiya et al., 1985], and [Katehakis & Veinott, 1987]. [Duff, 1995] provides a reinforcement learning approach that gradually learns indices through online/model-free interaction with bandit processes. The details of these algorithms would require more space than is available here. The algorithm proposed in the next section makes use of the approach of [Varaiya et al., 1985].

# 4 LOCAL BANDIT APPROXIMATION AND AN APPROXIMATELY-OPTIMAL LEARNING ALGORITHM

The most obvious difference between the optimal learning problem for an MDP and the multi-armed bandit problem is that the MDP has a phase-space component (Markov chain state) to its hyperstate. A first step in bandit-based approximation, then, proceeds by "removing" this phase-space component. This can be achieved by viewing the process on a time-scale defined by the recurrence time of a given state. That is, suppose the process is in some state, $x$. In response to some given action, two things can happen: (1) The process can transition, in one time-step, into $x$ again with some immediate reward, or (2) The process can transition into some state that is not $x$ and experience some "sojourn" path of states and rewards before returning to $x$. On a time-scale defined by sojourn-time, one can view the process in a sort of "state-$x$-centric" way (if state $x$ never recurs, then the sojourn-time is "infinite" and there is no value-of-information component of the local bandit model to acknowledge). From this perspective, the process appears to have only one state, and is *semi*-Markov; that is, the time between transitions is a random variable. Some other action taken in state $x$ would give rise to a different sojourn reward process. For both processes (sojourn-processes initiated by different actions applied to state $x$), the sojourn path/reward will depend upon the policy for states encountered along sojourn paths, but suppose that this policy is fixed for the moment. By viewing the original process on a time-scale of sojourn-time, one has effectively collapsed the phase-space component of the hyperstate. The new process has one state, $x$, and the problem of choosing an action, given that one is uncertain about the transition probabilities, presents itself as a semi-Markov bandit problem.

The preceding discussion suggests an algorithm for approximately-optimal learning:

(0) Given that the uncertainty in transition probabilities is expressed in terms of sufficient statistics $< \vec{\alpha}, \vec{\beta} >$, and the process is currently in state $x_t$.

(1) Compute the optimal policy for the mean process, $\pi^*[\bar{P}(\vec{\alpha}, \vec{\beta})]$; that is, compute the policy that is optimal for the MDP whose transition probabilities are taken to be the mean values associated with $< \vec{\alpha}, \vec{\beta} >$—this defines a nominal (certainty-equivalent) policy for sojourn states.

(2) Construct a local bandit model at state $x_t$; that is, the decision-maker must choose between some number (the number of admissible actions) of sojourn reward processes—this is a semi-Markov multi-armed bandit problem.

(3) Compute the Gittins indices for the local bandit model.

(4) Take the action with the largest index.

(5) Observe a transition to $x_{t+1}$ in the underlying MDP.

(6) Update $< \vec{\alpha}, \vec{\beta} >$ accordingly (Bayes update).

(7) Go to step (1)

The local semi-Markov bandit process associated with **state 1 / action 1** for the 2-state example MDP of Figure 1 is shown in Figure 2. The sufficient statistics for $p_{11}^1$ are denoted by $(\alpha, \beta)$, and $\frac{\alpha}{\alpha+\beta}$ and $\frac{\beta}{\alpha+\beta}$ are the expected probabilities for transition into **state 1** and **state 2**, respectively. $\Gamma$ and $R_{121}$ are random variables signifying sojourn time and reward.

The goal is to compute the index for the root information-state labeled $< \alpha, \beta >$ and to compare it with that computed for a similar diagram associated with the bandit

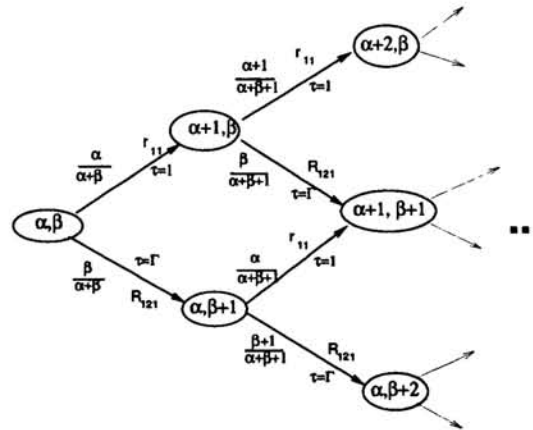

Figure 2: A local semi-Markov bandit process associated with **state 1 / action 1** for the 2-state example MDP of Figure 1.

process for taking **action 2**. The approximately-optimal action is suggested by the process having the largest root-node index. Indices for semi-Markov bandits can be obtained by considering the bandits as Markov, but performing the optimization in Equation 1 over a restricted set of stopping times. The algorithm suggested in [Tsitsiklis, 1993], which in turn makes use of methods described in [Varaiya *et al.*, 1985], proceeds by "reducing" the graph through a sequence of node-excisions and modifications of rewards and transition probabilities; [Duff, 1997] details how these steps may be realized for the special semi-Markov processes associated with problems of optimal learning.

## 5  Discussion

In summary, this paper has presented the problem of optimal learning, in which a decision-maker is obliged to enjoy or endure the consequences of its actions in quest of the asymptotically-learned optimal policy. A Bayesian formulation of the problem leads to a clear concept of a solution whose computation, however, appears to entail an examination of an intractably-large number of hyperstates. This paper has suggested extending the Gittins index approach (which applies with great power and elegance to the special class of multi-armed bandit processes) to general adaptive MDP's. The hope has been that if certain salient features of the value of information could be captured, even approximately, then one could be led to a reasonable method for avoiding certain defects of certainty-equivalence approaches (problems with identifiability, "metastability"). Obviously, positive evidence, in the form of empirical results from simulation experiments, would lend support to these ideas— work along these lines is underway.

Local bandit approximation is but one approximate computational approach for problems of optimal learning and dual control. Most prominent in the literature of control theory is the "wide-sense" approach of [Bar-Shalom & Tse, 1976], which utilizes local quadratic approximations about nominal state/control trajectories. For certain problems, this method has demonstrated superior performance compared to a certainty-equivalence approach, but it is computationally very intensive and unwieldy, particularly for problems with controller dimension greater than one.

One could revert to the view of the bandit problem, or general adaptive MDP, as simply a very large MDP defined over hyperstates, and then consider a some-

what direct approach in which one performs approximate dynamic programming with function approximation over this domain—details of function-approximation, feature-selection, and "training" all become important design issues. [Duff, 1997] provides further discussion of these topics, as well as a consideration of action-elimination procedures [MacQueen, 1966] that could result in substantial pruning of the hyperstate decision tree.

## Acknowledgements

This research was supported, in part, by the National Science Foundation under grant ECS-9214866 to Andrew G. Barto.

## References

Bar-Shalom, Y. & Tse, E. (1976) Caution, probing and the value of information in the control of uncertain systems, *Ann. Econ. Soc. Meas.* 5:323-337.

R. Bellman & R. Kalaba, (1959) On adaptive control processes. *IRE Trans.*, **4**:1-9.

Bokar, V. & Varaiya, P.P. (1979) Adaptive control of Markov chains I: finite parameter set. *IEEE Trans. Auto. Control* **24**:953-958.

Cozzolino, J.M., Gonzalez-Zubieta, R., & Miller, R.L. (1965) Markov decision processes with uncertain transition probabilities. *Tech. Rpt. 11, Operations Research Center, MIT.*

Dayan, P. & Sejnowski, T. (1996) Exploration Bonuses and Dual Control. *Machine Learning* (in press).

Duff, M.O. (1995) Q-learning for bandit problems. in *Machine Learning: Proceedings of the Twelfth International Conference on Machine Learning*: pp. 209-217.

Duff, M.O. (1997) Approximate computational methods for optimal learning and dual control. *Technical Report,* Deptartment of Computer Science, Univ. of Massachusetts, Amherst.

Fel'dbaum, A. (1965) *Optimal Control Systems,* Academic Press.

Gittins, J.C. & Jones, D. (1979) Bandit processes and dynamic allocation indices (with discussion). *J. R. Statist. Soc. B* **41**:148-177.

Katehakis, M.H. & Veinott, A.F. (1987) The multi-armed bandit problem: decomposition and computation *Math. OR* **12**: 262-268.

MacQueen, J. (1966). A modified dynamic programming method for Markov decision problems, *J. Math. Anal. Appl.,* 14:38-43.

Robinsion, D.R. (1981) Algorithms for evaluating the dynamic allocation index. *Research Report No. 80/DRR/4, Manchester-Sheffield School of Probability and Statistics.*

Tsitsiklis, J. (1993) A short proof of the Gittins index theorem. *Proc. 32nd Conf. Dec. and Control:* 389-390.

Varaiya, P.P., Walrand, J.C., & Buyukkoc, C. (1985) Extensions of the multiarmed bandit problem: the discounted case. *IEEE Trans. Auto. Control* **30**(5):426-439.

Watkins, C. (1989) *Learning from Delayed Rewards* Ph.D. Thesis, Cambidge University.